# A Smoothing Regularizer for Recurrent Neural Networks

**Lizhong Wu and John Moody**

Oregon Graduate Institute, Computer Science Dept., Portland, OR 97291-1000

## Abstract

We derive a smoothing regularizer for recurrent network models by requiring robustness in prediction performance to perturbations of the training data. The regularizer can be viewed as a generalization of the first order Tikhonov stabilizer to dynamic models. The closed-form expression of the regularizer covers both time-lagged and simultaneous recurrent nets, with feedforward nets and one-layer linear nets as special cases. We have successfully tested this regularizer in a number of case studies and found that it performs better than standard quadratic weight decay.

## 1 Introduction

One technique for preventing a neural network from overfitting noisy data is to add a regularizer to the error function being minimized. Regularizers typically smooth the fit to noisy data. Well-established techniques include ridge regression, see (Hoerl & Kennard 1970), and more generally spline smoothing functions or Tikhonov stabilizers that penalize the $m^{th}$-order squared derivatives of the function being fit, as in (Tikhonov & Arsenin 1977), (Eubank 1988), (Hastie & Tibshirani 1990) and (Wahba 1990). These methods have recently been extended to networks of radial basis functions (Girosi, Jones & Poggio 1995), and several heuristic approaches have been developed for sigmoidal neural networks, for example, quadratic weight decay (Plaut, Nowlan & Hinton 1986), weight elimination (Scalettar & Zee 1988),(Chauvin 1990),(Weigend, Rumelhart & Huberman 1990) and soft weight sharing (Nowlan & Hinton 1992).[1] All previous studies on regularization have concentrated on feedforward neural networks. To our knowledge, recurrent learning with regularization has not been reported before.

In Section 2 of this paper, we develop a smoothing regularizer for general dynamic models which is derived by considering perturbations of the training data. We present a closed-form expression for our regularizer for two layer feedforward and recurrent neural networks, with standard weight decay being a special case. In Section 3, we evaluate our regularizer's performance on predicting the U.S. Index of Industrial Production. The advantage of our regularizer is demonstrated by comparing to standard weight decay in both feedforward and recurrent modeling. Finally, we conclude our paper in Section 4.

## 2 Smoothing Regularization

### 2.1 Prediction Error for Perturbed Data Sets

Consider a training data set $\{P : Z(t), X(t)\}$, where the targets $Z(t)$ are assumed to be generated by an unknown dynamical system $F^*(I(t))$ and an unobserved noise process:

$$Z(t) = F^*(I(t)) + \varepsilon^*(t) \quad \text{with} \quad I(t) = \{X(s), s = 1, 2, \cdots, t\} \ . \tag{1}$$

Here, $I(t)$ is the information set containing both current and past inputs $X(s)$, and the $\varepsilon^*(t)$ are independent random noise variables with zero mean and variance $\sigma^{*2}$. Consider next a dynamic network model $\hat{Z}(t) = F(\Phi, I(t))$ to be trained on data set $P$, where $\Phi$ represents a set of network parameters, and $F(\ )$ is a network transfer function which is assumed to be nonlinear and dynamic. We assume that $F(\ )$ has good approximation capabilities, such that $F(\Phi_P, I(t)) \approx F^*(I(t))$ for learnable parameters $\Phi_P$.

Our goal is to derive a smoothing regularizer for a network trained on the actual data set $P$ that in effect optimizes the expected network performance (prediction risk) on perturbed test data sets of form $\{Q : \tilde{Z}(t), \tilde{X}(t)\}$. The elements of Q are related to the elements of P via small random perturbations $\varepsilon_z(t)$ and $\varepsilon_x(t)$, so that

$$\tilde{Z}(t) = Z(t) + \varepsilon_z(t), \tag{2}$$
$$\tilde{X}(t) = X(t) + \varepsilon_x(t). \tag{3}$$

The $\varepsilon_z(t)$ and $\varepsilon_x(t)$ have zero mean and variances $\sigma_z^2$ and $\sigma_x^2$ respectively. The training and test errors for the data sets $P$ and $Q$ are

$$D_P = \frac{1}{N} \sum_{t=1}^{N} [Z(t) - F(\Phi_P, I(t))]^2 \tag{4}$$

$$D_Q = \frac{1}{N} \sum_{t=1}^{N} [\tilde{Z}(t) - F(\Phi_P, \tilde{I}(t))]^2 \ , \tag{5}$$

where $\Phi_P$ denotes the network parameters obtained by training on data set $P$, and $\tilde{I}(t) = \{\tilde{X}(s), s = 1, 2, \cdots, t\}$ is the perturbed information set of $Q$. With this notation, our goal is to minimize the expected value of $D_Q$, while training on $D_P$.

Consider the prediction error for the perturbed data point at time $t$:

$$d(t) = [\tilde{Z}(t) - F(\Phi_P, \tilde{I}(t))]^2 \ . \tag{6}$$

With Eqn (2), we obtain

$$\begin{aligned} d(t) &= [Z(t) + \varepsilon_z(t) - F(\Phi_P, I(t)) + F(\Phi_P, I(t)) - F(\Phi_P, \tilde{I}(t))]^2, \\ &= [Z(t) - F(\Phi_P, I(t))]^2 + [F(\Phi_P, I(t)) - F(\Phi_P, \tilde{I}(t))]^2 + [\varepsilon_z(t)]^2 \\ &\quad + 2[Z(t) - F(\Phi_P, I(t))][F(\Phi_P, I(t)) - F(\Phi_P, \tilde{I}(t))] \\ &\quad + 2\varepsilon_z(t)[Z(t) - F(\Phi_P, \tilde{I}(t))]. \end{aligned} \tag{7}$$

Assuming that $\varepsilon_z(t)$ is uncorrelated with $[Z(t) - F(\Phi_P, \tilde{I}(t))]$ and averaging over the exemplars of data sets $P$ and $Q$, Eqn(7) becomes

$$
\begin{aligned}
D_Q &= D_P + \frac{1}{N}\sum_{t=1}^{N}[F(\Phi_P, I(t)) - F(\Phi_P, \tilde{I}(t))]^2 + \frac{1}{N}\sum_{t=1}^{N}[\varepsilon_z(t)]^2 \\
&+ \frac{2}{N}\sum_{t=1}^{N}[Z(t) - F(\Phi_P, I(t))][F(\Phi_P, I(t)) - F(\Phi_P, \tilde{I}(t))] .
\end{aligned} \tag{8}
$$

The third term, $\sum_{t=1}^{N}[\varepsilon_z(t)]^2$, in Eqn(8) is independent of the weights, so it can be neglected during the learning process. The fourth term in Eqn(8) is the cross-covariance between $[Z(t) - F(\Phi_P, I(t))]$ and $[F(\Phi_P, I(t)) - F(\Phi_P, \tilde{I}(t))]$. Using the inequality $2ab \le a^2 + b^2$, we can see that minimizing the first term $D_P$ and the second term $\frac{1}{N}\sum_{t=1}^{N}[F(\Phi_P, I(t)) - F(\Phi_P, \tilde{I}(t))]^2$ in Eqn (8) during training will automatically decrease the effect of the cross-covariance term. Therefore, we exclude the cross-covariance term from the training criterion.

The above analysis shows that the expected test error $D_Q$ can be minimized by minimizing the objective function $D$:

$$
D = \frac{1}{N}\sum_{t=1}^{N}[Z(t) - F(\Phi, I(t))]^2 + \frac{1}{N}\sum_{t=1}^{N}[F(\Phi_P, I(t)) - F(\Phi_P, \tilde{I}(t))]^2 . \tag{9}
$$

In Eqn (9), the second term is the time average of the squared disturbance $\|\hat{\tilde{Z}}(t) - \hat{Z}(t)\|^2$ of the trained network output due to the input perturbation $\|\tilde{I}(t) - I(t)\|^2$. Minimizing this term demands that small changes in the input variables yield correspondingly small changes in the output. This is the standard smoothness prior, namely that if nothing else is known about the function to be approximated, a good option is to assume a high degree of smoothness. Without knowing the correct functional form of the dynamical system $F^*$ or using such prior assumptions, the data fitting problem is ill-posed. In (Wu & Moody 1996), we have shown that the second term in Eqn (9) is a dynamic generalization of the first order Tikhonov stabilizer.

## 2.2   Form of the Proposed Smoothing Regularizer

Consider a general, two layer, nonlinear, dynamic network with recurrent connections on the internal layer [2] as described by

$$
Y(t) = \mathbf{f}\left(WY(t-\tau) + VX(t)\right), \hat{Z}(t) = UY(t) \tag{10}
$$

where $X(t)$, $Y(t)$ and $\hat{Z}(t)$ are respectively the network input vector, the hidden output vector and the network output; $\Phi = \{U, V, W\}$ is the output, input and recurrent connections of the network; $\mathbf{f}(\ )$ is the vector-valued nonlinear transfer function of the hidden units; and $\tau$ is a time delay in the feedback connections of hidden layer which is pre-defined by a user and will not be changed during learning. $\tau$ can be zero, a fraction, or an integer, but we are interested in the cases with a small $\tau$.[3]

When $\tau = 1$, our model is a recurrent network as described by (Elman 1990) and (Rumelhart, Hinton & Williams 1986) (see Figure 17 on page 355). When $\tau$ is equal to some fraction smaller than one, the network evolves $\frac{1}{\tau}$ times within each input time interval. When $\tau$ decreases and approaches zero, our model is the same as the network studied by (Pineda 1989), and earlier, widely-studied *additive networks*. In (Pineda 1989), $\tau$ was referred to as the *network relaxation time scale*. (Werbos 1992) distinguished the recurrent networks with zero $\tau$ and non-zero $\tau$ by calling them *simultaneous recurrent networks* and *time-lagged recurrent networks* respectively.

We have found that minimizing the second term of Eqn(9) can be obtained by smoothing the output response to an input perturbation at every time step. This yields, see (Wu & Moody 1996):

$$\|\tilde{\hat{Z}}(t) - \hat{Z}(t)\|^2 \leq \rho_\tau{}^2(\Phi_P)\|\tilde{X}(t) - X(t)\|^2 \quad \text{for } t = 1, 2, \ldots, N \ . \tag{11}$$

We call $\rho_\tau{}^2(\Phi_P)$ the *output sensitivity* of the trained network $\Phi_P$ to an input perturbation. $\rho_\tau{}^2(\Phi_P)$ is determined by the network parameters only and is independent of the time variable $t$.

We obtain our new regularizer by training directly on the expected prediction error for perturbed data sets $Q$. Based on the analysis leading to Eqns (9) and (11), the training criterion thus becomes

$$D = \frac{1}{N}\sum_{t=1}^{N}[Z(t) - F(\Phi, I(t))]^2 + \lambda\rho_\tau{}^2(\Phi) \ . \tag{12}$$

The coefficient $\lambda$ in Eqn(12) is a regularization parameter that measures the degree of input perturbation $\|\tilde{I}(t) - I(t)\|^2$. The algebraic form for $\rho_\tau(\Phi)$ as derived in (Wu & Moody 1996) is:

$$\rho_\tau(\Phi) = \frac{\gamma\|U\|\|V\|}{1 - \gamma\|W\|}\left\{1 - \exp\left(\frac{\gamma\|W\| - 1}{\tau}\right)\right\} \ , \tag{13}$$

for time-lagged recurrent networks ($\tau > 0$). Here, $\|\ \|$ denotes the Euclidean matrix norm. The factor $\gamma$ depends upon the maximal value of the first derivatives of the activation functions of the hidden units and is given by:

$$\gamma = \max_{t,j}|f_j{}'(o_j(t))| \ , \tag{14}$$

where $j$ is the index of hidden units and $o_j(t)$ is the input to the $j^{th}$ unit. In general, $\gamma \leq 1$. [4] To insure stability and that the effects of small input perturbations are damped out, it is required, see (Wu & Moody 1996), that

$$\gamma\|W\| < 1 \ . \tag{15}$$

The regularizer Eqn(13) can be deduced for the simultaneous recurrent networks in the limit $\tau \mapsto 0$ by:

$$\rho(\Phi) \equiv \rho_0(\Phi) = \frac{\gamma\|U\|\|V\|}{1 - \gamma\|W\|} \ . \tag{16}$$

If the network is feedforward, $W = 0$ and $\tau = 0$, Eqns (13) and (16) become

$$\rho(\Phi) = \gamma\|U\|\|V\| \ . \tag{17}$$

Moreover, if there is no hidden layer and the inputs are directly connected to the outputs via $U$, the network is an ordinary linear model, and we obtain

$$\rho(\Phi) = \|U\| \ , \tag{18}$$

which is standard quadratic weight decay (Plaut *et al.* 1986) as is used in ridge regression (Hoerl & Kennard 1970).

The regularizer (Eqn(17) for feedforward networks and Eqn (13) for recurrent networks) was obtained by requiring smoothness of the network output to perturbations of data. We therefore refer to it as a smoothing regularizer. Several approaches can be applied to estimate the regularization parameter $\lambda$, as in (Eubank 1988), (Hastie & Tibshirani 1990) and (Wahba 1990). We will not discuss this subject in this paper.

In the next section, we evaluate the new regularizer for the task of predicting the U.S. Index of Industrial Production. Additional empirical tests can be found in (Wu & Moody 1996).

## 3   Predicting the U.S. Index of Industrial Production

The Index of Industrial Production (IP) is one of the key measures of economic activity. It is computed and published monthly. Our task is to predict the one-month rate of change of the index from January 1980 to December 1989 for models trained from January 1950 to December 1979. The exogenous inputs we have used include 8 time series such as the index of leading indicators, housing starts, the money supply M2, the *S&P* 500 Index. These 8 series are also recorded monthly. In previous studies by (Moody, Levin & Rehfuss 1993), with the same defined training and test data sets, the normalized prediction errors of the one month rate of change were 0.81 with the **neuz** neural network simulator, and 0.75 with the **proj** neural network simulator.

We have simulated feedforward and recurrent neural network models. Both models consist of two layers. There are 9 input units in the recurrent model, which receive the 8 exogenous series and the previous month IP index change. We set the time-delayed length in the recurrent connections $\tau = 1$. The feedforward model is constructed with 36 input units, which receive 4 time-delayed versions of each input series. The time-delay lengths are 1, 3, 6 and 12, respectively. The activation functions of hidden units in both feedforward and recurrent models are *tanh* functions. The number of hidden units varies from 2 to 6. Each model has one linear output unit.

We have divided the data from January 1950 to December 1979 into four non-overlapping sub-sets. One sub-set consists of 70% of the original data and each of the other three subsets consists of 10% of the original data. The larger sub-set is used as training data and the three smaller sub-sets are used as validation data. These three validation data sets are respectively used for determination of early stopped training, selecting the regularization parameter and selecting the number of hidden units.

We have formed 10 random training-validation partitions. For each training-validation partition, three networks with different initial weight parameters are trained. Therefore, our prediction committee is formed by 30 networks.

The committee error is the average of the errors of all committee members. All networks in the committee are trained simultaneously and stopped at the same time based on the committee error of a validation set. The value of the regularization parameter and the number of hidden units are determined by minimizing the committee error on separate validation sets.

Table 1 compares the out-of-sample performance of recurrent networks and feedfor-

Table 1: Normalized prediction errors for the one-month rate of return on the U.S. Index of Industrial Production (Jan. 1980 - Dec. 1989). Each result is based on 30 networks.

| Model | Regularizer | Mean ± Std | Median | Max | Min | Committee |
|---|---|---|---|---|---|---|
| Recurrent | Smoothing | 0.646±0.008 | 0.647 | 0.657 | 0.632 | 0.639 |
| Networks | Weight Decay | 0.734±0.018 | 0.737 | 0.767 | 0.704 | 0.734 |
| Feedforward | Smoothing | 0.700±0.023 | 0.707 | 0.729 | 0.654 | 0.693 |
| Networks | Weight Decay | 0.745±0.043 | 0.748 | 0.805 | 0.676 | 0.731 |

ward networks trained with our smoothing regularizer to that of networks trained with standard weight decay. The results are based on 30 networks. As shown, the smoothing regularizer again outperforms standard weight decay with 95% confidence (in $t$-distribution hypothesis) in both cases of recurrent networks and feedforward networks. We also list the median, maximal and minimal prediction errors over 30 predictors. The last column gives the committee results, which are based on the simple average of 30 network predictions. We see that the median, maximal and minimal values and the committee results obtained with the smoothing regularizer are all smaller than those obtained with standard weight decay, in both recurrent and feedforward network models.

## 4  Concluding Remarks

Regularization in learning can prevent a network from overtraining. Several techniques have been developed in recent years, but all these are specialized for feedforward networks. To our best knowledge, a regularizer for a recurrent network has not been reported previously.

We have developed a smoothing regularizer for recurrent neural networks that captures the dependencies of input, output, and feedback weight values on each other. The regularizer covers both simultaneous and time-lagged recurrent networks, with feedforward networks and single layer, linear networks as special cases. Our smoothing regularizer for linear networks has the same form as standard weight decay. The regularizer developed depends on only the network parameters, and can easily be used. A more detailed description of this work appears in (Wu & Moody 1996).

## Footnotes

[1] Two additional papers related to ours, but dealing only with feed forward networks, came to our attention or were written after our work was completed. These are (Bishop 1995) and (Leen 1995). Also, Moody & Rögnvaldsson (1995) have recently proposed several new classes of smoothing regularizers for feedforward nets.

[2] Our derivation can easily be extended to other network structures.

[3] When the time delay $\tau$ exceeds some critical value, a recurrent network becomes unstable and lies in oscillatory modes. See, for example, (Marcus & Westervelt 1989).

[4]For instance, $f'(x) = [1 - f(x)]f(x)$ if $f(x) = \frac{1}{1+e^{-x}}$. Then, $\gamma = \max|f'(x)| = \frac{1}{4}$.

## References

Bishop, C. (1995), 'Training with noise is equivalent to Tikhonov regularization', *Neural Computation* 7(1), 108–116.

Chauvin, Y. (1990), Dynamic behavior of constrained back-propagation networks, *in* D. Touretzky, ed., 'Advances in Neural Information Processing Systems 2', Morgan Kaufmann Publishers, San Francisco, CA, pp. 642–649.

Elman, J. (1990), 'Finding structure in time', *Cognition Science* 14, 179–211.

Eubank, R. L. (1988), *Spline Smoothing and Nonparametric Regression*, Marcel Dekker, Inc.

Girosi, F., Jones, M. & Poggio, T. (1995), 'Regularization theory and neural networks architectures', *Neural Computation* 7, 219–269.

Hastie, T. J. & Tibshirani, R. J. (1990), *Generalized Additive Models*, Vol. 43 of *Monographs on Statistics and Applied Probability*, Chapman and Hall.

Hoerl, A. & Kennard, R. (1970), 'Ridge regression: biased estimation for nonorthogonal problems', *Technometrics* **12**, 55–67.

Leen, T. (1995), 'From data distributions to regularization in invariant learning', *Neural Computation* **7**(5), 974–981.

Marcus, C. & Westervelt, R. (1989), Dynamics of analog neural networks with time delay, *in* D. Touretzky, ed., 'Advances in Neural Information Processing Systems 1', Morgan Kaufmann Publishers, San Francisco, CA.

Moody, J. & Rögnvaldsson, T. (1995), Smoothing regularizers for feed-forward neural networks, Oregon Graduate Institute Computer Science Dept. Technical Report, submitted for publication, 1995.

Moody, J., Levin, U. & Rehfuss, S. (1993), 'Predicting the U.S. index of industrial production', *In proceedings of the 1993 Parallel Applications in Statistics and Economics Conference, Zeist, The Netherlands. Special issue of* Neural Network World **3**(6), 791–794.

Nowlan, S. & Hinton, G. (1992), 'Simplifying neural networks by soft weight-sharing', *Neural Computation* **4**(4), 473–493.

Pineda, F. (1989), 'Recurrent backpropagation and the dynamical approach to adaptive neural computation', *Neural Computation* **1**(2), 161–172.

Plaut, D., Nowlan, S. & Hinton, G. (1986), Experiments on learning by back propagation, Technical Report CMU-CS-86-126, Carnegie-Mellon University.

Rumelhart, D., Hinton, G. & Williams, R. (1986), Learning internal representations by error propagation, *in* D. Rumelhart & J. McClelland, eds, 'Parallel Distributed Processing: Exploration in the microstructure of cognition', MIT Press, Cambridge, MA, chapter 8, pp. 319–362.

Scalettar, R. & Zee, A. (1988), Emergence of grandmother memory in feed forward networks: learning with noise and forgetfulness, *in* D. Waltz & J. Feldman, eds, 'Connectionist Models and Their Implications: Readings from Cognitive Science', Ablex Pub. Corp.

Tikhonov, A. N. & Arsenin, V. I. (1977), *Solutions of Ill-posed Problems*, Winston ; New York : distributed solely by Halsted Press. Scripta series in mathematics. Translation editor, Fritz John.

Wahba, G. (1990), *Spline models for observational data*, CBMS-NSF Regional Conference Series in Applied Mathematics.

Weigend, A., Rumelhart, D. & Huberman, B. (1990), Back-propagation, weight-elimination and time series prediction, *in* T. Sejnowski, G. Hinton & D. Touretzky, eds, 'Proceedings of the connectionist models summer school', Morgan Kaufmann Publishers, San Mateo, CA, pp. 105–116.

Werbos, P. (1992), Neurocontrol and supervised learning: An overview and evaluation, *in* D. White & D. Sofge, eds, 'Handbook of Intelligent Control', Van Nostrand Reinhold, New York.

Wu, L. & Moody, J. (1996), 'A smoothing regularizer for feedforward and recurrent neural networks', *Neural Computation* **8**(3), 463–491.